# Learning to Solve Quadratic Unconstrained Binary Optimization in a Classification Way

**Ming Chen**[1]*, **Jie Chun** [1]*, **Shang Xiang** [2], **Luona Wei**[3], **Yonghao Du**[1], **Qian Wan**[4],
**Yuning Chen**[1]†, **Yingwu Chen**[1]

[1]College of Systems Engineering, National University of Defense Technology
[2]School of Public Administration, Xiangtan University
[3]College of Electronics and Information Engineering, South-Central Minzu University
[4]National Engineering Research Center of Educational Big Data,
Central China Normal University
{cmself, chunjie0720, xiangshang165, wlnelysion, duyonghao15,
wanq8228}@163.com, cynnudt@hotmail.com, cywnudt@163.com

## Abstract

The quadratic unconstrained binary optimization (QUBO) is a well-known NP-hard problem that takes an $n \times n$ matrix $Q$ as input and decides an $n$-dimensional 0-1 vector $x$, to optimize a quadratic function. Existing learning-based models that always formulate the solution process as sequential decisions suffer from high computational overload. To overcome this issue, we propose a neural solver called the Value Classification Model (VCM) that formulates the solution process from a classification perspective. It applies a Depth Value Network (DVN) based on graph convolution that exploits the symmetry property in $Q$ to auto-grasp value features. These features are then fed into a Value Classification Network (VCN) which directly generates classification solutions. Trained by a highly efficient model-tailored Greedy-guided Self Trainer (GST) which does not require any priori optimal labels, VCM significantly outperforms competitors in both computational efficiency and solution quality with a remarkable generalization ability. It can achieve near-optimal solutions in milliseconds with an average optimality gap of just 0.362% on benchmarks with up to 2500 variables. Notably, a VCM trained at a specific DVN depth can steadily find better solutions by simply extending the testing depth, which narrows the gap to 0.034% on benchmarks. To our knowledge, this is the first learning-based model to reach such a performance.

## 1 Introduction

Nonlinear integer programming is a highly challenging subject in mathematical programming and operations research, where the Quadratic Unconstrained Binary Optimization (QUBO) problem is one of the most well-known cases. Due to its extensive applicability and computational intricacies, the QUBO continues to receive widespread attention[1]. The purpose of the QUBO problem is to optimize an unconstrained quadratic function:

$$\max / \min OFV = f(x) = x^\top Q x = \sum_{i=1}^{n} \sum_{j=1}^{n} q_{ij} x_i x_j \qquad (1)$$

where $Q$ is a symmetric matrix with $n \times n$ coefficients, while $x$ is a binary (0-1) $n$-dimensional column vector, i.e., $x_i \in \{0, 1\}, i = 1, ..., n$. $OFV$ is short for objective function value.

This simple formulation is able to represent a remarkable spectrum of applications in Combinatorial Optimization (CO), including those from quantum computing [1], asset exchange problem [2], financial analysis [3], transaction settlement [4], set packing problem [5], linear ordering problem [6], etc. A large number of classic graph problems and constrained quadratic problems can be re-casted into QUBO through simple transformations [7, 8, 9, 10], for which QUBO solving methods can be easily applied. From a computational perspective, the QUBO problem belongs to the NP-hard family. Typical solving methods for the QUBO include exact methods and heuristic methods. Due to its NP-hardness, the QUBO is not expected to be solved by any exact method in polynomial time. For this reason, intensive research has been devoted to developing heuristic methods.

In recent years, the research on learning-based Neural CO (NCO) methods for solving CO problems has become a hot topic in this field [11, 12]. This type of data-driven method has strong scalability and high efficiency for problems of different types and scales, as opposed to traditional search-based metaheuristics which often require substantial computational effort when problem size is large. These works typically model the solution of CO problems as a Markov Decision Process (MDP) [13], and then use deep reinforcement learning (DRL) to train an advanced policy network, Pointer Network (PN) [14] and Graph Neural Network (GNN)-based [15] models for instance, to support dynamic decision-making at each MDP step. Despite the remarkable performance of these models in addressing certain classical CO problems such as the Vehicle Routing Problem (VRP) [16], their applications to QUBO encounter significant limitations:

**1) PN-based DRL models.** PN-based models require problem data represented by a $n \times C$ matrix to satisfy its fixed $C$ embedding channels, where $n$ is the number of tasks and $C$ is a fixed number of features per task. For instance, the Travelling Salesman Problem (TSP) can be represented by $n$ nodes and their coordinates (each node described by an $(x, y)$ pair), forming an $n \times 2$ matrix. Due to such a requirement, PN-based models encounter difficulties in processing $n \times n$-dimensional input matrix of the QUBO directly since it is not able to handle problem data where both dimensions are variable, resulting in weak scalability of the model for problems of different scales.

**2) GNN-based DRL models.** Conversely, GNN-based models [15] excel at encoding $n \times n$ graph data by graph embeddings. Despite this advantage, GNNs incur significant computational and storage costs due to the repeated updating of node and edge hidden states at each MDP step, particularly for large problem sizes. For instance, the GNN in [15] requires an $n \times n \times h$-dimensional hidden matrices to store edge encodings, which is $n$ times the storage of the PN needed at the same hidden size $h$. Our preliminary tests revealed that the machine we used for this work could not handle the storage demands of GNNs for large-scale instances (see the results of P7000 in Table 1). Additionally, [17] reported that GNNs are not able to compete with greedy heuristics in certain graph combinatorial optimization problems, aligning with our QUBO experimental results.

The limitations encountered by these networks in solving QUBO are closely related to the MDP-solving paradigm attached to them. When addressing QUBO via an MDP, each step involves flipping the assigned value of a selected binary variable (see the top of Figure 1). According to [18], it is critical to evaluate the impact of each flip on the $OFV$ to ensure the high quality of the resulting solution. However, the repeated $OFV$ evaluation throughout the MDP leads to significant computational overhead, which may be unacceptable for large-scale instances.

**Contributions.** Given the above limitations, a radical improvement would require revolutionizing the solution paradigm to compensate for the limitations of DRL models by not using MDP and avoiding repeated $OFV$ computations. Considering the binary nature of QUBO variables, a novel solution paradigm based on classification is proposed, which can provide the binary values of all variables directly at once, rather than following sequential decision-making to determine values one by one.

To operationalize the concept of this innovative QUBO classification solution paradigm, we propose a Value Classification Model (VCM)[3], which is a neural solver comprised of three key components: an Extractor, the Depth Value Network (DVN), for feature extraction; a Classifier, the Value Classification Network (VCN), for solution determination; and a Trainer, the Greedy-guided Self Trainer (GST), for solver training. Based on graph convolutional, DVN exploits the symmetry property of input data of QUBO to efficiently extract the value features, with performance steadily improving as depth increases. These resulting value features of each variable enable VCN to directly generate the solutions in a classification way. For effective model training, we propose the GST, which does not

need any prior optimal labels. Extensive experiments demonstrate the effectiveness of the proposed VCM and GST. A well-trained VCM can directly generate near-optimal solutions for QUBO within milliseconds and exhibit remarkable generalization capabilities across both instance sizes and data distributions. For example, a VCM trained on instances of size 10 can produce near-optimal results for instances of size 7,000 in milliseconds. Furthermore, simply by increasing the testing DVN depth, a VCM trained at a specific depth can attain even better solutions. To our knowledge, this represents the first learning-based classification model for solving QUBO with such a performance.

## 2 Related Works

The literature on QUBO dates back to [19], which introduced pseudo-boolean functions and binary quadratic optimization. Due to the combinatorial complexity, QUBO has been proven NP-hard in 1979 [20]. Over the decades, a large number of methods for solving QUBO have emerged in the literature. The research on exact methods primarily employed branch-and-bound [21, 22, 23, 24, 25] or branch-and-cut [23]. Since the exact methods are prohibitively expensive when applied to large-size instances, various heuristic methods have extensively emerged, which are designed for high-quality solution discovery within acceptable timeframes. Noteworthy ones include Tabu Search (TS) [26, 27, 28], Simulated Annealing (SA) [29], Local Search (LS) [30], Ant Colony Optimization (ACO) [31], Memetic Algorithm (MA) [32] and hybrid algorithms [33].

Recently, learning-based neural CO methods have exhibited significant promise in addressing CO challenges. Successful applications include 3-D Bin Packing Problem [34], Maximum Cut [15, 35], Routing Problems [14, 36, 37, 38, 39, 16, 40, 41], etc. Unlike traditional search-based methods, these approaches aim to autonomously acquire solution policies from a significant amount of problem data. Nevertheless, the integration of these learning-based heuristics into QUBO-specific applications is still in its infancy, with only a handful of studies exploring this direction and they all adopted the solving paradigm based on MDP. Based on PN, [18] designed hand-crafted features, such as row sum, diagonal elements, and $OFV$ values. These feature values can only reflect a small number of characteristics of the problem data, making it difficult to characterize the overall picture of the problem, leading to unsatisfactory results. In addition, since this method models the problem solution process as an MDP, it requires repeated calculation of $OFV$ values, resulting in low solving efficiency.

## 3 The Neural Solver VCM

**General scheme.** The general scheme of our proposed Value Classification Model (VCM) is outlined in Figure 1 at the bottom, with the detailed neural architecture provided in Appendix A. The neural solver VCM comprises three key components: an Extractor, the Depth Value Network (DVN), for feature extraction; a Classifier, the Value Classification Network (VCN), for solution determination; and a Trainer, the Greedy-guided Self Trainer (GST), for solver training.

**The Extractor DVN.** The Extractor is responsible for automatically grasping variables and their correlation features from $Q$ matrix to support the Classifier. The efficacy of the Extractor essentially determines the performance of the VCM. We notice that the $Q$ matrix is graph-like data. It is known that the Graph Convolutional Network (GCN) [42] is a robust technique for graph representation, which learns hidden layer representations that encode both local graph structure and features of nodes. The extraction of the $l$-th layer is summarized as $H^{(l+1)} = \sigma(\tilde{D}^{-1/2}(A + I_N)\tilde{D}^{-1/2}H^{(l)}W^{(l)})$. Here, $A$, $I_N$, and $\tilde{D}$ are the adjacency matrix, identity matrix, and the degree matrix of the undirected graph $\mathcal{G}$. $W^{(l)}$ is a layer-specific trainable weight matrix, and $H^{(l)}$ is the matrix of activations in the $l$-th layer. The core operation in graph convolution is $AH$. In QUBO, the $A$ is instantiated as $Q$, while $H$ is the set of extracted features of the variable vector $x$.

However, directly using GCN as the Extractor for QUBO poses several insurmountable challenges. First, transforming QUBO into a graph typically results in a fully connected $Q$ matrix, leading to $n^2$ real-valued edges, with each edge $q_{ij}$ directly affecting the $OFV$ of the problem. It is essential to manage the extraction of large-scale edges and discern their distinctions. As stated in [43], the graph convolution in GCNs is a special form of Laplacian smoothing, a key factor for their effectiveness. This smoothing could diminish the influence of high-weight edges, which is not required in QUBO since their original weight differences are essential for the model to accurately identify their impact on the objective function. Also, GCN performance significantly degrades when its depth exceeds

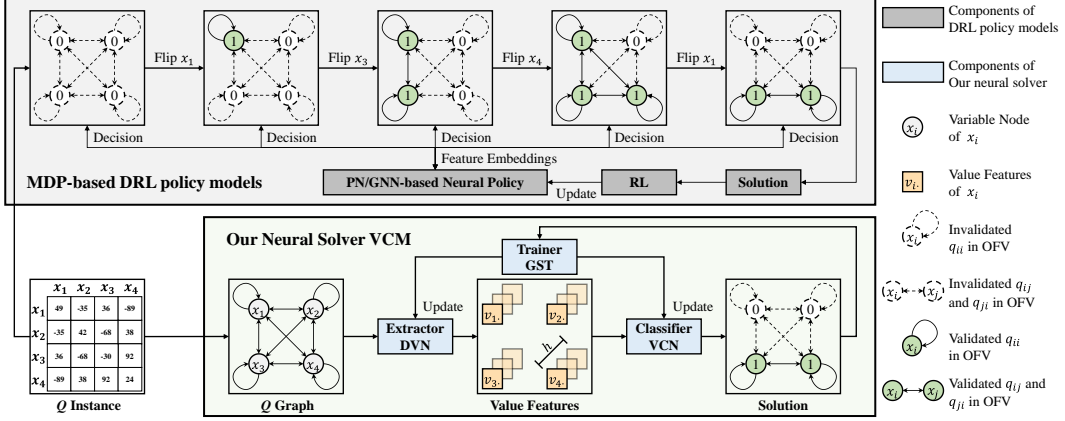

Figure 1: The processes of DRL-based policy models and our QUBO solver. Both PN and GNN-based DRL policy models construct solutions sequentially by capturing environmental embeddings at each step to support decision-making. In contrast, our neural solver VCM outputs solutions directly in a classification way, without any additional decision steps.

two layers [42], even when employing residual connections. This is because when a GCN contains numerous convolutional layers, the resulting output features may become excessively smoothed, making it difficult to differentiate vertices from various clusters. To address these issues and based on the principles of graph convolution, we propose a QUBO-tailored Depth Value Network (DVN).

In QUBO, each variable $x_j$ directly affects the elements of the corresponding $j$-th row and column in $Q$. Given that $Q$ is symmetric, i.e., $Q = Q^T$, the features associated with each row should be unified with those of the corresponding column. We first define the features of each variable $x_j$ to be extracted as a value feature, accompanied by a value feature vector $\boldsymbol{v}_j \in \mathbb{R}^h$, where $h$ denotes the hidden size. Let $\boldsymbol{v}_j$ be the value features of the $j$-th column of $Q$ (denoted as $\boldsymbol{v}_j^{col}$). Each element in row $j$ can then be expressed as $q_{ij}\boldsymbol{v}_j^{col}$. Consequently, the value features for the $i$-th row can be computed as $\boldsymbol{v}_i^{row} = \sum_{j=1}^n (q_{ij}\boldsymbol{v}_j^{col})$. To ensure unified row and column features, the value of each column, $\boldsymbol{v}_i^{col}$, should match the value of its corresponding row, $\boldsymbol{v}_i^{row}$. This requirement translates to unifying $\boldsymbol{V}$ and $Q\boldsymbol{V}$, where the computation of $Q\boldsymbol{V}$ mirrors the core operation of GCN. This alignment may be an additional reason why GCN exhibits effective performance.

In DVN, we take the $Q\boldsymbol{V}$ and current $\boldsymbol{V}$ as inputs and employ a learning function $\mathcal{F}_E$ to iterate the value feature. The iteration at depth $d$ is updated as follows:

$$\boldsymbol{V}^{(d+1)} = \mathcal{F}_E(\boldsymbol{V}^{(d)}, Q\boldsymbol{V}^{(d)}) \qquad (2)$$

Here, $\boldsymbol{V} = \{\boldsymbol{v}_i\}_{i=1}^N$ is the value feature matrix for all variables $x$ (i.e., the nodes in the graph). Each $\boldsymbol{v}_i \in \mathbb{R}^{h \times 1}$ is the value feature vector for $x_i$ with all initial values set to 1. $\mathcal{F}_E$ consists of specific neural networks and activation functions tailored for pattern recognition and features processing. Specifically, it obtains the new value features $\boldsymbol{V}^{(d),D}$ based on the current and $Q\boldsymbol{V}$ convolution value features as follows:

$$\boldsymbol{V}^{(d),D} = \boldsymbol{M}_3[\boldsymbol{V}^{(d)}; \tanh(\boldsymbol{M}_2(\text{ReLU}(\boldsymbol{M}_1 Q\boldsymbol{V}^{(d)})))] \qquad (3)$$

Where $\boldsymbol{M}_1, \boldsymbol{M}_2 \in \mathbb{R}^{h \times h}$ are learnable memory units that capture specific feature interactions. ReLU and tanh are the activation functions used to filter the extracted features and compress the value features within a limited threshold, respectively. [;] is the horizontal concatenation operator and $\boldsymbol{M}_3 \in \mathbb{R}^{h \times 2h}$ is a learnable memory unit that connects the current and convolution value features. Finally, $\boldsymbol{V}^{(d+1)}$ is then obtained as follows.

$$\boldsymbol{V}^{(d+1)} = \tanh(\boldsymbol{V}^{(d),D}) \qquad (4)$$

The output value of tanh, within a symmetric range of $[-1, 1]$, can effectively represent the features of distinct variables. An ablation study on activation functions and memory units demonstrates the

effectiveness of our neural architecture design (see Appendix E). These filtering and compression processes allow DVN to iteratively extract features from Q at any depth, which effectively avoids the problem of decreased performance caused by the convolutional layer increase of GCN. Additionally, during data initialization, we scale the data by $\lambda Q$ to enhance the efficiency of value compression (see Appendix A), where $\lambda$ represents the scaling factor with a specific value. Experimental results verify that the performance of our solver steadily improves when the iteration depth of the DVN increases. Notably, without incurring additional training costs, our solver can find better solutions simply by extending the iteration depth (see Section 4.3).

**The Classifier VCN.** Based on the obtained value features $\boldsymbol{V}^{(d),D}$, we propose a Value Classification Network (VCN) which serves as the classifier to generate the solution $x$ for the QUBO.

$$x = \mathcal{F}_C(\boldsymbol{V}^{(d),D}) \tag{5}$$

Where $\mathcal{F}_C$ is a learning function that maps the value feature of each variable into a binary value. To this end, we calculate the state of each $x_i$ and use the activation function $\tanh$ to obtain the state of each $x_i$. The classification result is then determined based on the state of $x_i$.

$$\{state_i\}_{i=1}^n = \tanh(\boldsymbol{u}\tanh(\boldsymbol{M}_4\boldsymbol{V}^{(d),D})) \tag{6}$$

$$x_i = \left\{ \begin{array}{ll} 1, & state_i > 0 \\ 0, & state_i \leq 0 \end{array} \right. \tag{7}$$

Where $\boldsymbol{M}_4 \in \mathbb{R}^{h \times h}$ is a learnable memory unit and $\boldsymbol{u} \in \mathbb{R}^{1 \times h}$ is the learnable uniform vector to integrate the value features of each variable. VCN can produce a complete solution directly by simultaneously considering all variables through a single classification action. Compared to other DRL policy models that rely on sequential decisions, our solver can significantly reduce the computational complexity from $O(kn^2)$ (where the sequence length $k$ can potentially expand to $2^n$ when the flip of all variables is enumerated) to $O(n^2)$.

**The Trainer GST.** It is known that the classification model usually requires labeled solutions for training. Unfortunately, in the case of QUBO, acquiring optimal solutions is rather expensive and may be infeasible for large-size problems. For this reason, we propose a Greedy-guided Self Trainer (GST), which effectively avoids the need for pre-labeled optimal solutions. The GST is outlined in Algorithm 1 and its working logic is shown in Figure 2, which consists of three components: a VCM, a Batch Greedy Flip (BGF) algorithm, and a historical best solution set (HB) $X^L$, where the BGF and HB are two featured ingredients of the GST.

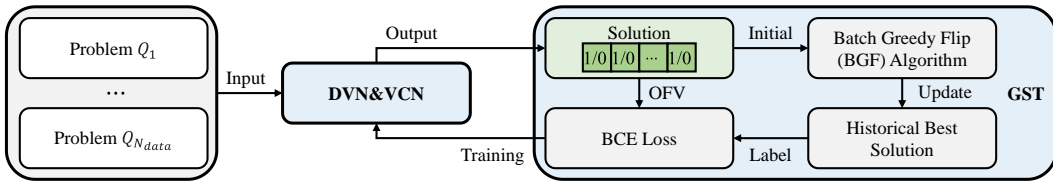

Figure 2: Working logic of the GST.

The Greedy Flip algorithm employs a flip operation to change the assignment for each variable $x_i$ between 0 and 1 (detailed in Appendix B.1). At each step, the algorithm first calculates the Objective Function Value Increment (OFI) for each variable, which quantifies the change of $OFV$ resulting from flipping $x_i$ (detailed in Appendix B.2). The flip operation with the highest positive OFI is selected. The above process continues until all variables are processed. However, computing the OFI at each step is computationally expensive, making it impractical for batch training.

To overcome this computational bottleneck, we propose a batch OFI calculation technique (detailed in Appendix B) that leverages matrix computation acceleration to significantly enhance efficiency. Based on this, we develop a Batch Greedy Flip (BGF) method (detailed in Appendix C), a GPU-accelerated heuristic that improves on the VCM (termed VCM-BGF) by identifying and correcting obvious sub-optimal classification in the current VCM solution.

**Algorithm 1** The Greedy-guided Self Trainer
***
**Input:** The training dataset $D$ with $N_{data}$ instances, the epoch size $E$, the batch size $B$, the training steps per epoch $N_{steps} = ceil(N_{data}/B)$
**Output:** The trained VCM with parameters $\theta^*$
Initialize the VCM parameters $\theta$, Adam optimizer and the label set $X_{steps}^L = X_1^L \cup ... \cup X_{N_{steps}}^L$
**for** $epoch = 1$ **to** $E$ **do**
    **for** $step = 1$ **to** $N_{steps}$ **do**
        $data_{step} \leftarrow SampleInput(D)$
        $X_{step}^L \leftarrow SampleInput(X_{steps}^L)$
        $state_{step}, X_{step}^{VCM} \leftarrow VCM(data_{step})$
        $X_{step}^G \leftarrow BatchGreedyFlip(X_{step}^{VCM}, data_{step})$
        $X_{step}^L \leftarrow \arg\max_{X \in \{X_{step}^G, X_{step}^L\}} OFV(X)$
        $Loss \leftarrow BCELoss(X_{step}^L, (state_{step} + 1)/2)$
        $\theta \leftarrow Adam(\theta, Loss)$
    **end for**
**end for**
***

Finally, we introduce a historical best solution set $X^L$. During the multi-epoch training process, the training dataset is fixed, ensuring that each instance is used once in each training epoch. This allows us to use the VCM-BGF solution $X^G$ at each epoch to maintain and update the historical best solution set $X^L$. The updated historical best solutions serve as the training labels for our neural solver. This VCM-based, data-driven label-generating process yields adaptive, high-quality labels at low cost. The Binary Cross Entropy (BCE) is applied to calculate the training loss, with the optimization handled by the well-known Adam optimizer [44].

## 4 Experiments

### 4.1 Experimental Details

**Datasets.** The datasets used in our experiments include generated instances (G), benchmarks (B), and well-known instances (P), described in the format: dataset+instance size+(number of instances). For the G set, the $Q$ matrix is uniformly generated at random within [-100,100], following the benchmark data format. The B set is B2500(10) consisting of ten ORLIB instances of size 2500 [45]. The P set includes 21 very-large instances [46] including P3000(5), P4000(5), P5000(5), P6000(3), and P7000(3). The average $OFV$ gap to the current optimal baseline (GAP in %) and the average running time (ART in milliseconds for default, ms) are used as the evaluation indicators.

**Parameter setting.** Following the Parameters Study (see Appendix F), the default values for VCM parameters are set as $h = 128$, $\alpha = 4$, and $d = 40$. VCM is trained for 100 epochs under four sets of small-size instances (with 10, 20, 50, and 100 variables) with a batch size of 512, resulting in 400 VCMs. Each set includes 512,000 G instances (limited by memory). For fair validation, the test batch size is fixed at 1. Each model is initialized with Xavier initialization [47], and the Adam optimizer is applied with a $10^{-4}$ learning rate and 0.975 decay factor. Experiments were run on an NVIDIA GeForce RTX 3090 and an Intel i9-9900K CPU with 64GB RAM and Ubuntu 18.04 using Pytorch 1.90 in a Python 3.7 environment.

**Competitors.** Our evaluation includes several types of competitors: 1) The exact optimizer Gurobi [48]. We set the max allowed time to 1s and 1h. 2) Heuristic classification methods, Diag and SR, proposed in [18]. 3) Heuristic construction algorithm BGF, part of GST. 4) The physics-inspired neural solver, PI-GNN [49], with varying numbers of layers. 5) Three learning-based sequential-decision construction models: one PN-based model called DRLH [18] (which is the state-of-the-art), two GNN-based models called S2V-DQN [15] and ECO-DQN [35] (which are the most relevant state-of-the-art models for solving optimization problems over graphs). These models use the same parameter setting as VCM and are accelerated by our batch OFI calculation technique, resulting in competitors DRLH-B, S2V-DQN-B, and ECO-DQN-B. To assess the stability of VCM, we also include its enhanced version VCM-BGF as a competitor. In addition, for each instance of the datasets, we obtain a high-quality reference solution using an integrated model (called VCM-BGF-HB)

composed of 400 trained VCM-BGFs under depth $d = 100$. VCM-BGF-HB is deemed a high-quality baseline since it is able to achieve an average 0.012% deviation from benchmark optimality (see Table 1).

## 4.2 Experimental Results

We use the B set and P set to validate the performance of the trained VCMs. The results in Table 1 show that PN-based DRLH-B requires seconds to obtain suboptimal solutions. In terms of solution quality, DRLH-B is outperformed by the ECO-DQN-B, yet it incurs substantial time increases and computational costs due to graph embedding, resulting in an insurmountable GPU memory limitation to preclude their execution on P7000. Among the learning-based competitors, the 2-layer PI-GNN demonstrates the best performance in terms of solution quality. Interestingly, our proposed BGF easily outperforms these learning-based models in both solution quality and speed. Yet it is still dominated by the VCM which is the best solver. Indeed, it surpasses all competitors across the whole instance set in both quality and computational efficiency with the highest Wilcoxon P-value of 3.09E-03 (see Appendix D). Taking VCM50 as an example, it can achieve near-optimal solutions with an average gap of only 0.362% within 8ms for benchmarks. Such a solution speed is rather impressive. The results of the very large instances from the P set show that our proposed VCM maintains near-optimal performance. Particularly, the VCM trained on instances with 10 variables achieves an average gap of 0.569% on P7000, displaying significant generalization ability. The results of the VCM-BGF-HB and Gurobi are also presented as a reference to the VCM. They produce optimal or near-optimal solutions. However, Gurobi can only solve problems in a sequential way and incurs substantial time cost (over 1h for each G100 instance in Appendix G).

Table 1: Results on benchmarks and large well-known instances.

| ALGORITHM | B2500(10) | | P3000(5) | | P4000(5) | | P5000(5) | | P6000(3) | | P7000(3) | |
|---|---|---|---|---|---|---|---|---|---|---|---|---|
| | OBJECTIVE FUNCTION VALUE BASELINE (OPTIMAL) | | | | | | | | | | | |
| | 1479921.4 | | 5134727.2 | | 7869134.2 | | 10973791.4 | | 13950582.0 | | 17725010.33 | |
| | GAP (%) | ART (MS) | GAP (%) | ART (MS) | GAP (%) | ART (MS) | GAP (%) | ART (MS) | GAP (%) | ART (MS) | GAP (%) | ART (MS) |
| DIAG | 81.842 | 0.7 | 99.262 | 1.7 | 97.374 | 2.9 | 98.256 | 4.4 | 99.273 | 11.1 | 98.843 | 14.3 |
| SR | 26.867 | 0.1 | 32.861 | 0.0 | 33.510 | 0.1 | 33.428 | 0.2 | 33.679 | 0.0 | 32.758 | 0.4 |
| VCM10 | 0.368 | 7.8 | **0.566** | 7.9 | **0.440** | 8.1 | **0.508** | 8.5 | **0.645** | 8.8 | **0.569** | 9.1 |
| VCM20 | 0.488 | 7.7 | 0.598 | 7.9 | 0.673 | 8.1 | 0.657 | 8.3 | 0.712 | 8.6 | 0.646 | 9.1 |
| VCM50 | **0.362** | 7.7 | 0.861 | 7.9 | 0.669 | 8.1 | 0.783 | 8.1 | 0.806 | 8.7 | 0.702 | 9.1 |
| VCM100 | 0.401 | 7.8 | 0.763 | 7.9 | 0.668 | 8.0 | 0.803 | 8.0 | 0.914 | 8.7 | 0.772 | 9.1 |
| BGF | 0.807 | 800.8 | 0.979 | 984.4 | 0.834 | 999.4 | 0.914 | 994.7 | 0.966 | 999.9 | 0.730 | 983.6 |
| DRLH-B | 1.640 | 1.5E+03 | 2.044 | 2.4E+03 | 1.884 | 5.0E+03 | 1.853 | 8.9E+03 | 2.030 | 1.5E+04 | 1.825 | 2.2E+04 |
| S2V-DQN-B | 21.657 | 1.6E+04 | 13.172 | 2.7E+04 | 13.255 | 6.3E+04 | 14.050 | 1.2E+05 | 14.403 | 2.1E+05 | - | - |
| ECO-DQN-B | 0.937 | 2.8E+04 | 1.371 | 5.4E+04 | 1.333 | 1.2E+05 | 1.270 | 2.3E+05 | 1.467 | 4.0E+05 | - | - |
| VCM10-BGF | 0.230 | 40.8 | 0.382 | 70.6 | 0.297 | 107.4 | 0.322 | 196.3 | 0.374 | 444.4 | 0.335 | 613.6 |
| VCM20-BGF | 0.227 | 52.3 | 0.260 | 87.8 | 0.342 | 135.8 | 0.310 | 282.6 | 0.316 | 506.8 | 0.312 | 634.7 |
| VCM50-BGF | **0.136** | 44.1 | 0.277 | 100.2 | 0.214 | 170.0 | 0.262 | 326.0 | **0.267** | 523.3 | 0.224 | 770.2 |
| VCM100-BGF | 0.139 | 49.2 | **0.203** | 106.0 | **0.178** | 186.7 | **0.245** | 298.2 | 0.271 | 549.7 | **0.212** | 770.4 |
| PI-GNN(2-LAYER) | 1.689 | 4.4E+04 | 2.13 | 6.2E+04 | 1.636 | 7.1E+04 | 1.418 | 8.6E+04 | 1.986 | 1.1E+05 | 1.437 | 1.6E+05 |
| PI-GNN(3-LAYER) | 1.909 | 5.7E+04 | 2.523 | 7.2E+04 | 2.092 | 8.0E+04 | 1.945 | 1.0E+05 | 2.18 | 1.3E+05 | 2.076 | 1.9E+05 |
| PI-GNN(5-LAYER) | 3.28 | 1.2E+05 | 3.047 | 1.0E+05 | 2.761 | 1.1E+05 | 2.463 | 1.3E+05 | 2.584 | 1.8E+05 | 2.289 | 2.7E+05 |
| GUROBI-1S | 0.034 | 1.0E+03 | 0.070 | 1.0E+03 | 0.108 | 1.0E+03 | 0.091 | 1.0E+03 | 0.122 | 1.0E+03 | 0.145 | 1.0E+03 |
| GUROBI-1H | **0.0023** | 3.6E+06 | **0.0028** | 3.6E+06 | **0.0109** | 3.6E+06 | **0.0096** | 3.6E+06 | **0.0144** | 3.6E+06 | **0.0169** | 3.6E+06 |
| VCM-BGF-HB | 0.012 | 1.9E+04 | 0.020 | 3.6E+04 | 0.027 | 6.0E+04 | 0.040 | 1.1E+05 | 0.030 | 2.0E+05 | 0.057 | 2.8E+05 |

[1] The **Bold** indicates the best average result in different classes of methods.

To further investigate the generalization ability of the VCM, we conduct experiments on an additional 1,000 generated G instances with 20, 50, 100, 200, 500, and 1,000 variables. We use VCM-BGF-HB as the optimal baseline. The results, illustrated in Figure 3 and detailed in Appendix G, demonstrate that a VCM trained at a specific size performs well on instances of other sizes. For example, the gap between VCM10 and VCM100 on G100 instances is merely 0.056%, demonstrating the remarkable generalization ability of VCM. The results also show that BGF can further enhance the performance of the VCM, with a solution quality improvement of 0.27% on average. Meanwhile, this improvement afforded by BGF is limited, reflecting the inherent stability of VCM.

In comparison to MDP-based methods, VCM replaces this complex sequential decision-making process with a simple classification process, providing an inherent advantage in solving efficiency that becomes more pronounced as the instance size increases. Notably, the VCM achieves this performance without using OFI (which is essential in sequential decision-based methods), thereby

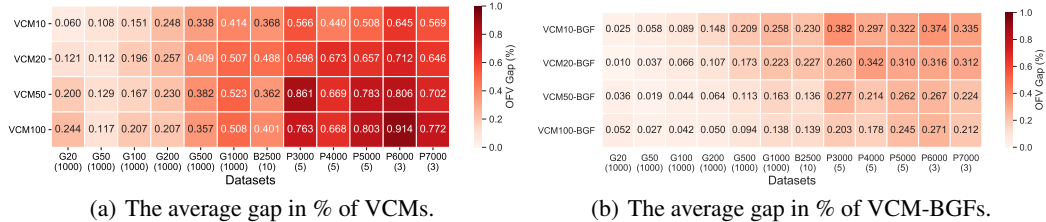

(a) The average gap in % of VCMs.        (b) The average gap in % of VCM-BGFs.

Figure 3: The size generalization ability of VCM and VCM-BGF under different datasets.

considerably reducing computational overhead. The computing time is thousands of times less on average. Also, the computing time increases moderately as the problem size enlarges (see Figure 4). Across the whole datasets (with variables ranging from 20 to 7000), the computing time is always under 10ms. In contrast, other sequential-decision competitors, especially GNN-based models, exhibit excessive time growth as the problem size grows, since the number of decision-making steps increases significantly.

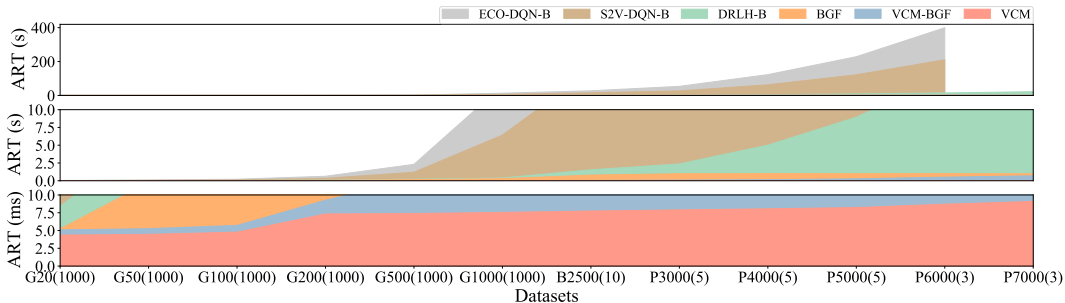

Figure 4: Avearge running time of different datasets in seconds and milliseconds.

## 4.3 Model Study

**Efficiency of GST.** We introduce three training competitors, UnS [49], LHB, and LGF, to validate our Trainer GST. The UnS is an unsupervised training method that directly uses the optimization objective function as the loss function and relaxes the 0-1 variables for optimization. The labels for LHB are obtained by VCM-BGF-HB. The LHB can be considered supervised learning with optimal labels. The labels for LGF are obtained using the current VCM-BGF, which is GST without the historical best solution set. We adopt VCM-BGF-HB as the optimal baseline and illustrate the training curves in Figure 5. The results indicate that GST outperforms competitors in terms of both efficiency and stability. Specifically, GST demonstrates a more efficient and stable training process compared to the unsupervised trainer UnS, which suffers from considerable fluctuations. GST achieves the same performance as the supervised LHB in the early stages of the training process while requiring at least 50% fewer epochs. Although LGF can reach local optimum from time to time, its training process experiences fluctuations due to the unstable quality of the labels outputted by VCM-BGF alone. Therefore, the integration of BGF and historical solutions within GST enables a rapid, stable, and adaptive formation of VCM, and circumvents the substantial costs associated with supervised learning.

**Distribution Generalization of VCM.** The G set, whose default distribution is denoted as R-1, with all elements of the matrix non-zero. To assess the generalization ability of VCM across diverse data distributions, we generate new datasets by following a standard normal random distribution (RN-1) and deactivating the matrix elements to 0 with probabilities of 10% (R-0.9), 40% (R-0.6), 70% (R-0.3), and 90% (R-0.1). We conduct tests on these instances of diverse distributions with 20, 50, and 100 variables using our trained corresponding-sized VCMs. For each distribution, we generate 1,000 G instances and apply VCM-BGF-HB as the optimal baseline to measure the gap. The results (summarized in Figure 6 and detailed in Appendix H) demonstrate that VCM maintains

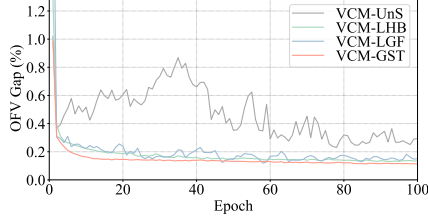

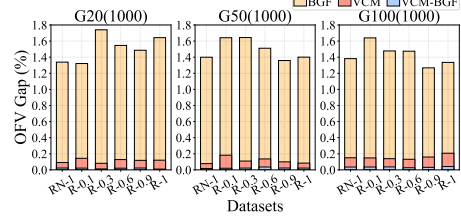

Figure 5: VCM training process under different methods at instance size 50.

Figure 6: VCM results under different distribution instances.

near-optimal across various data distributions. The VCM performance on the R-0.1 instance set with 100 variables, where 90% Q matrix elements are zero, manages to sustain a performance gap no larger than 0.15%. Besides, since the MaxCut problem can be re-casted into UQBO, with the difference in the data distribution, we extend our evaluation to MaxCut benchmarks to validate our findings. Results detailed in Appendix I further confirm the practical applicability of VCM. These observations imply that the Extractor in VCM can be considered a generic module with distribution-independent and problem-specific in QUBO, thereby bringing VCM a remarkable generalization ability and wide-spread applicability.

**The Study of DVN Depth.** The depth is crucial for ensuring interaction and coherence between row and column features in DVN graph convolution. As shown in the Parameter Study in Appendix F.2, increasing the training depth of DVN improves feature extraction precision, leading to consistent enhancement in VCM performance. It is logical to hypothesize that a VCM trained at a particular depth $d$ would perform better at deeper testing depths. To investigate this hypothesis, we evaluate six VCMs on benchmark B2500 across eight testing depths. The results (illustrated in Figure 7(a) and detailed in Appendix J) show that all VCMs exhibit significant performance gains as testing depth increases. Even at the training depth of 10, VCM already learns an iterative feature extraction pattern. However, when the testing depth retracts below the training depth, VCM presents a perceptible but tolerable performance degradation, reinforcing the premise that reaching the training depth threshold is essential for DVN stabilization.

To further investigate the rationale behind this performance, we apply the t-SNE [50] on the output $\boldsymbol{V}^{(d),D}$ of Extractor DVN from the trained VCM50-$d$10 with varying testing depths. Figure 8 shows that as $d$ increases, clustering transitions into binary clusters gradually. When $d = 1$, it converges to a number of lines, and when $d > 100$, the binary clustering occurs. This demonstrates that our DVN overcomes the limitation of GCN (i.e., its performance degrades as the depth increases, as shown in our evaluation in Appendix K), thus effectively supporting the Classifier VCN. For instance, the default VCM50-$d$40 achieves an average gap of merely 0.034% when the testing $d$ hits 300, as opposed to its original gap of 0.362%. This performance enhancement can be generalizable to very large instances, as demonstrated by our evaluations on well-known instances (see Appendix J) and G instance with 10,000 and 20,000 variables (see Appendix L), and is achieved without additional training costs. As shown in Figure 7(b), the computational time increases linearly (on average 0.17ms per depth) as the testing depth enlarges, which verifies the remarkable performance of the VCM and extends its applications.

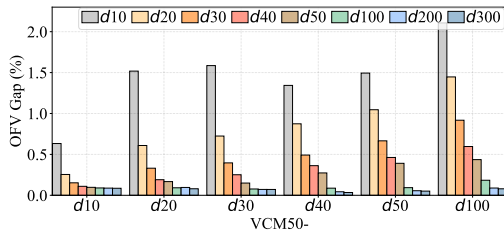

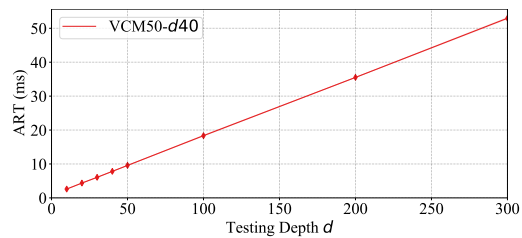

(a) Different testing depth $d$ on B2500(10).

(b) Computational time of VCM50-$d$40 on B2500(10).

Figure 7: VCM results under different testing depth $d$.

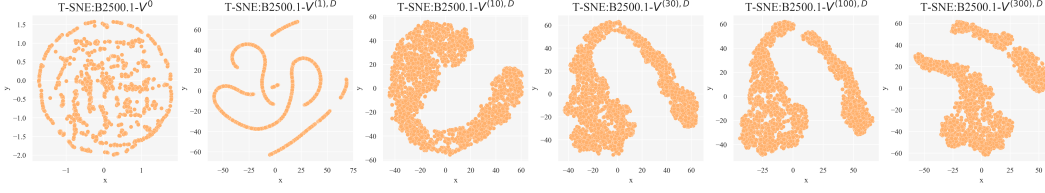

Figure 8: The t-SNE visualizations of $\boldsymbol{V}^{(d),D}$ on the first instance of B2500 under VCM50-$d$10.

# 5 Conclusions and Future Work

Our neural solver VCM is a new state-of-the-art learning-based model designed for efficiently solving QUBO from a classification perspective. It applies the Extractor DVN based on graph convolution and exploits the symmetry property in $Q$ to auto-grasp value features. Utilizing these resultant value features of each variable, VCM generates solutions directly through the Classifier VCN in a classification way. Trained by a highly efficient model-tailored Trainer GST which does not require any priori optimal labels, VCM shows near-optimality, high efficiency, and generalization ability in problem-solving. In particular, it can achieve an average gap to benchmark optimality of 0.362% in milliseconds and steadily narrow it down to 0.034% by simply extending the testing depth, which brings only a linearly increased computational time (average 0.17ms per depth).

Although VCM reaches such a performance by single classification decision-making, its optimality may be potentially limited by the VCN utilization of value features from DVN. This limitation poses challenges in achieving the optimal solution. Future research could explore enhancing VCM capabilities, including the development of more accurate classification networks and more efficient trainers, as well as adapting it to address other combinatorial optimization problems and real-world applications.

## Acknowledgments and Disclosure of Funding

This work was supported in part by the National Natural Science Foundation of China (No. 72271240, 72201272, 62307015 and 62437002); in part by the Natural Science Foundation of Hunan Province (No. 2022JJ30671 and 2024JJ4047); in part by the Hubei Provincial Natural Science Foundation of China (No. 2024AFB169, 2023AFB295 and 2023AFA020); in part by the China Postdoctoral Science Foundation (No. 2023M741304); and in part by the Scientific Research Foundation of Hunan Provincial Education Department (No. 22C0050).

## Footnotes

†Correspondence

[3]The code is available at `https://github.com/cmself100/VCM-QUBO`.

## References

[1] Fred Glover, Gary Kochenberger, Rick Hennig, and Yu Du. Quantum bridge analytics i: a tutorial on formulating and using qubo models. *Annals of Operations Research*, 314(1):141–183, 2022.

[2] Fred Glover, Gary Kochenberger, Moses Ma, and Yu Du. Quantum bridge analytics ii: Qubo-plus, network optimization and combinatorial chaining for asset exchange. *Annals of Operations Research*, 314(1):185–212, 2022.

[3] Seo Woo Hong, Pierre Miasnikof, Roy Kwon, and Yuri Lawryshyn. Market graph clustering via qubo and digital annealing. *Journal of Risk and Financial Management*, 14(1):34, 2021.

[4] Lee Braine;Daniel J. Egger;Jennifer Glick;Stefan Woerner. Quantum algorithms for mixed binary optimization applied to transaction settlement. *IEEE Transactions on Quantum Engineering*, pages 1–8, 2021.

[5] Bahram Alidaee, Gary Kochenberger, Karen Lewis, Mark Lewis, and Haibo Wang. A new approach for modeling and solving set packing problems. *European Journal of Operational Research*, 186(2):504–512, 2008.

[6] Mark Lewis, Bahram Alidaee, Fred Glover, and Gary Kochenberger. A note on xqx as a modelling and solution framework for the linear ordering problem. *International Journal of Operational Research*, 5(2):152–162, 2009.

[7] Iain Dunning, Swati Gupta, and John Silberholz. What works best when? a systematic evaluation of heuristics for max-cut and qubo. *INFORMS Journal on Computing*, 30(3):608–624, 2018.

[8] Gary Kochenberger, Jin-Kao Hao, Fred Glover, Mark Lewis, Zhipeng Lü, Haibo Wang, and Yang Wang. The unconstrained binary quadratic programming problem: a survey. *Journal of combinatorial optimization*, 28:58–81, 2014.

[9] Endre Boros and Peter L Hammer. Pseudo-boolean optimization. *Discrete applied mathematics*, 123(1-3):155–225, 2002.

[10] Pierre Hansen. Methods of nonlinear 0-1 programming. In *Annals of Discrete Mathematics*, volume 5, pages 53–70. Elsevier, 1979.

[11] Cong Zhang, Yaoxin Wu, Yining Ma, Wen Song, Zhang Le, Zhiguang Cao, and Jie Zhang. A review on learning to solve combinatorial optimisation problems in manufacturing. *IET Collaborative Intelligent Manufacturing*, 5(1):e12072, 2023.

[12] Zhixiao Xiong, Fangyu Zong, Huigen Ye, and Hua Xu. Neuralqp: A general hypergraph-based optimization framework for large-scale qcqps, 2024.

[13] Richard S Sutton and Andrew G Barto. Reinforcement learning: An introduction. *Robotica*, 17(2):229–235, 1999.

[14] Oriol Vinyals, Meire Fortunato, and Navdeep Jaitly. Pointer networks. *Advances in neural information processing systems*, 28, 2015.

[15] Elias Khalil, Hanjun Dai, Yuyu Zhang, Bistra Dilkina, and Le Song. Learning combinatorial optimization algorithms over graphs. *Advances in neural information processing systems*, 30, 2017.

[16] Jianan Zhou, Yaoxin Wu, Wen Song, Zhiguang Cao, and Jie Zhang. Towards omni-generalizable neural methods for vehicle routing problems. *arXiv preprint arXiv:2305.19587*, 2023.

[17] Maria Chiara Angelini and Federico Ricci-Tersenghi. Modern graph neural networks do worse than classical greedy algorithms in solving combinatorial optimization problems like maximum independent set. *Nature Machine Intelligence*, 5(1):29–31, 2023.

[18] Ming Chen, Yuning Chen, Yonghao Du, Luona Wei, and Yingwu Chen. Heuristic algorithms based on deep reinforcement learning for quadratic unconstrained binary optimization. *Knowledge-Based Systems*, 207:106366, 2020.

[19] Peter L Hammer and Sergiu Rudeanu. Pseudo-boolean programming. *Operations Research*, 17(2):233–261, 1969.

[20] Panos M Pardalos and Somesh Jha. Complexity of uniqueness and local search in quadratic 0–1 programming. *Operations research letters*, 11(2):119–123, 1992.

[21] VP Gulati, SK Gupta, and AK Mittal. Unconstrained quadratic bivalent programming problem. *European Journal of Operational Research*, 15(1):121–125, 1984.

[22] Francisco Barahona, Michael Jünger, and Gerhard Reinelt. Experiments in quadratic 0–1 programming. *Mathematical Programming*, 44(1-3):127–137, 1989.

[23] Christoph Helmberg and Franz Rendl. Solving quadratic (0, 1)-problems by semidefinite programs and cutting planes. *Mathematical programming*, 82:291–315, 1998.

[24] Panos M Pardalos, Oleg A Prokopyev, and Stanislav Busygin. Continuous approaches for solving discrete optimization problems. *Handbook on modelling for discrete optimization*, pages 39–60, 2006.

[25] Hong-Xuan Huang, Panos M Pardalos, and Oleg A Prokopyev. Lower bound improvement and forcing rule for quadratic binary programming. *Computational Optimization and Applications*, 33:187–208, 2006.

[26] Yang Wang, Zhipeng Lü, Fred Glover, and Jin-Kao Hao. Effective variable fixing and scoring strategies for binary quadratic programming. In *Evolutionary Computation in Combinatorial Optimization: 11th European Conference, EvoCOP 2011, Torino, Italy, April 27-29, 2011. Proceedings 11*, pages 72–83. Springer, 2011.

[27] Yang Wang, Zhipeng Lü, Fred Glover, and Jin-Kao Hao. Backbone guided tabu search for solving the ubqp problem. *Journal of Heuristics*, 19:679–695, 2013.

[28] Jialong Shi, Qingfu Zhang, Bilel Derbel, and Arnaud Liefooghe. A parallel tabu search for the unconstrained binary quadratic programming problem. In *2017 IEEE Congress on Evolutionary Computation (CEC)*, pages 557–564. IEEE, 2017.

[29] Kengo Katayama and Hiroyuki Narihisa. Performance of simulated annealing-based heuristic for the unconstrained binary quadratic programming problem. *European Journal of Operational Research*, 134(1):103–119, 2001.

[30] Arnaud Liefooghe, Sébastien Verel, Luis Paquete, and Jin-Kao Hao. Experiments on local search for bi-objective unconstrained binary quadratic programming. In *International Conference on Evolutionary Multi-Criterion Optimization*, pages 171–186. Springer, 2015.

[31] Murilo Zangari, Aurora Pozo, Roberto Santana, and Alexander Mendiburu. A decomposition-based binary aco algorithm for the multiobjective ubqp. *Neurocomputing*, 246(JUL.12):58–68, 2017.

[32] Zhipeng Lü, Jin-Kao Hao, and Fred Glover. A study of memetic search with multi-parent combination for ubqp. In *Evolutionary Computation in Combinatorial Optimization: 10th European Conference, EvoCOP 2010, Istanbul, Turkey, April 7-9, 2010. Proceedings 10*, pages 154–165. Springer, 2010.

[33] Yang Wang, Zhipeng Lü, Fred Glover, and Jin-Kao Hao. Probabilistic grasp-tabu search algorithms for the ubqp problem. *Computers and Operations Research*, 40:3100–3107, 12 2013.

[34] Yuan Jiang, Zhiguang Cao, and Jie Zhang. Learning to solve 3-d bin packing problem via deep reinforcement learning and constraint programming. *IEEE transactions on cybernetics*, 2021.

[35] Thomas Barrett, William Clements, Jakob Foerster, and Alex Lvovsky. Exploratory combinatorial optimization with reinforcement learning. In *Proceedings of the AAAI conference on artificial intelligence*, volume 34, pages 3243–3250, 2020.

[36] Irwan Bello, Hieu Pham, Quoc V Le, Mohammad Norouzi, and Samy Bengio. Neural combinatorial optimization with reinforcement learning. *International Conference on Learning Representations*, 2017.

[37] Wouter Kool, Herke van Hoof, and Max Welling. Attention, learn to solve routing problems! *International Conference on Learning Representations*, 2019.

[38] Yeong-Dae Kwon, Jinho Choo, Byoungjip Kim, Iljoo Yoon, Youngjune Gwon, and Seungjai Min. Pomo: Policy optimization with multiple optima for reinforcement learning. *Advances in Neural Information Processing Systems*, 33:21188–21198, 2020.

[39] Jieyi Bi, Yining Ma, Jiahai Wang, Zhiguang Cao, Jinbiao Chen, Yuan Sun, and Yeow Meng Chee. Learning generalizable models for vehicle routing problems via knowledge distillation. *Advances in Neural Information Processing Systems*, 35:31226–31238, 2022.

[40] Fu Luo, Xi Lin, Fei Liu, Qingfu Zhang, and Zhenkun Wang. Neural combinatorial optimization with heavy decoder: Toward large scale generalization. *Advances in Neural Information Processing Systems*, 36, 2024.

[41] Jinbiao Chen, Zizhen Zhang, Zhiguang Cao, Yaoxin Wu, Yining Ma, Te Ye, and Jiahai Wang. Neural multi-objective combinatorial optimization with diversity enhancement. *Advances in Neural Information Processing Systems*, 36, 2024.

[42] Thomas N. Kipf and Max Welling. Semi-supervised classification with graph convolutional networks. In *International Conference on Learning Representations*, 2017.

[43] Qimai Li, Zhichao Han, and Xiao-Ming Wu. Deeper insights into graph convolutional networks for semi-supervised learning. In *Proceedings of the AAAI conference on artificial intelligence*, volume 32, 2018.

[44] D Kinga, Jimmy Ba Adam, et al. A method for stochastic optimization. In *International conference on learning representations (ICLR)*, volume 5, page 6. San Diego, California;, 2015.

[45] John E Beasley. Obtaining test problems via internet. *Journal of Global Optimization*, 8:429–433, 1996.

[46] Gintaras Palubeckis. Multistart tabu search strategies for the unconstrained binary quadratic optimization problem. *Annals of Operations Research*, 131:259–282, 2004.

[47] Xavier Glorot and Yoshua Bengio. Understanding the difficulty of training deep feedforward neural networks. In *Proceedings of the thirteenth international conference on artificial intelligence and statistics*, pages 249–256. JMLR Workshop and Conference Proceedings, 2010.

[48] LLC Gurobi Optimization. Gurobi optimizer reference manual, 2024.

[49] Martin JA Schuetz, J Kyle Brubaker, and Helmut G Katzgraber. Combinatorial optimization with physics-inspired graph neural networks. *Nature Machine Intelligence*, 4(4):367–377, 2022.

[50] Laurens Van der Maaten and Geoffrey Hinton. Visualizing data using t-sne. *Journal of machine learning research*, 9(11), 2008.

[51] Ankur Nath and Alan Kuhnle. A benchmark for maximum cut: Towards standardization of the evaluation of learned heuristics for combinatorial optimization. *arXiv preprint arXiv:2406.11897*, 2024.

# A The Neural Architecture within VCM

The overall neural architecture of solver VCM is plotted in Figure 9.

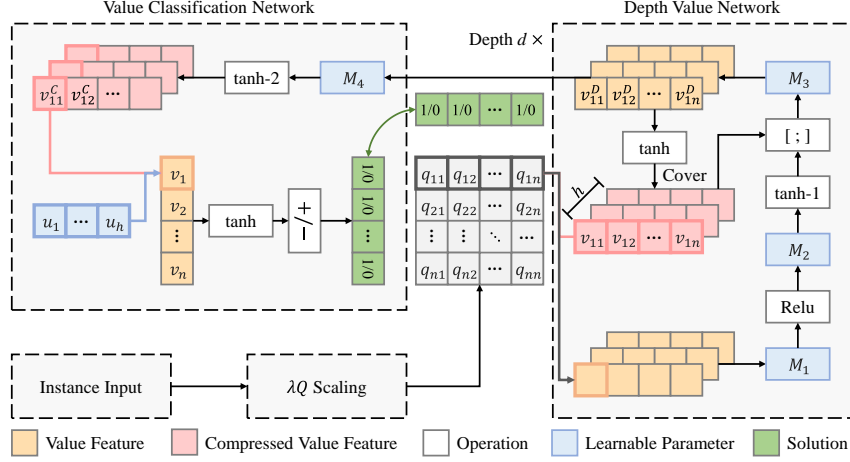

Figure 9: The neural architecture within VCM. The input Q is initially scaled by a scaling factor $\lambda$ to improve value compression efficiency for the post-order Extractor DVN. Subsequently, DVN utilizes problem-tailored convolution to iteratively auto-grasp value features for each variable based on the specified iterative depth $d$ and hidden size $h$. Finally, leveraging the acquired value features, the Classifier VCN generates the solution directly through a classification way.

Initially, in the Extractor DVN in VCM, the value feature $V$ of tasks is denoted as $V^0$ with elements initialized to 1, so that all elements of the first $QV^0$ represent the sum of $Q$'s row $rs$. These values are significantly influenced by the instance size and dataset distribution, which may vary in magnitude. Therefore, preprocessing of the input data is crucial. To enhance value compression efficiency in DVN, we scale $Q$ using a scaling function. For each $Q$, we take $\lambda Q$ to perform uniform scaling, where $\lambda$ is the scaling factor calculated as:

$$s = \mathrm{argmax}(\mathrm{abs}(rs_1, rs_2, ..., rs_n)) \tag{8}$$

$$\lambda = \alpha/(s + s_B) \tag{9}$$

Here, $\alpha$ is a handcrafted scaling constant, and $s_B$ is the mean of $s$ in each training batch. The proposed scaling function proves effective and plays a significant role in the performance of VCM (refer to the experimental results in Appendix F).

# B Batch Calculation of Objective Function Increment

## B.1 The Flip Operation

In the MDP for QUBO, the flip operation is fundamental, wherein a binary variable within the current solution transforms between 0 and 1 based on a specific strategy. The diagram of the flip operation is shown in Figure 10. The key basis for performing the flip operation is the calculation of the Objective Function Value Increment (OFI) after each variable is flipped, which is also an essential element of iterative search algorithms.

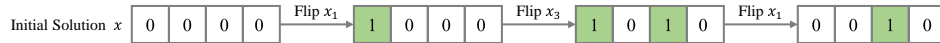

Figure 10: The flip operation in the MDP of QUBO.

## B.2 Batch OFI Calculation Method

Traditional OFI computation, however, as depicted on the left side of Figure 11, suffers from inefficiency and resource wastage due to its instance-by-instance serial execution. To address the computational inefficiencies inherent in the traditional OFI computation and to accommodate batch instance requirements pivotal for the existing deep learning model, we propose an innovative batch OFI calculation method. As depicted on the right side of Figure 11, the batch OFI calculation allows for simultaneous OFI calculations across $n$ variables in $B$ instances.

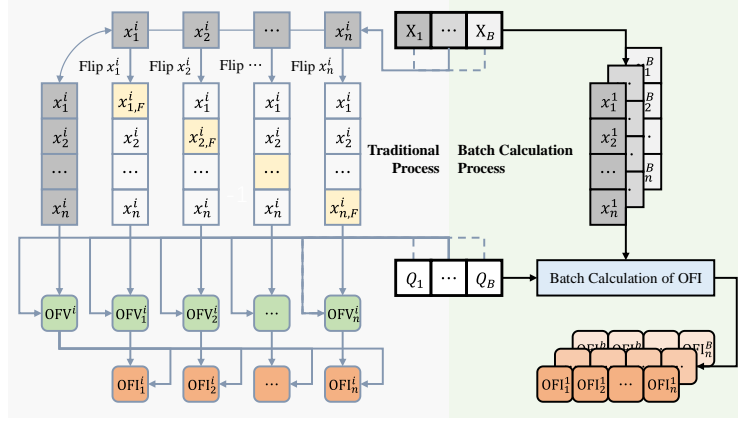

Figure 11: The OFI calculation process at each decision step of the traditional process and the proposed batch process in $B$ instances. The traditional process entails a sequential, instance-by-instance approach in which the OFI for each variable $x_i$ is serially and independently calculated at every decision step. In contrast, our proposed batch calculation process exploits the matrix computation to simplify and speed up the calculation process, enabling the determination of the OFI for $n$ variables across batch $B$ instances, with the potential for further acceleration through GPU.

We utilize matrix computation, presenting a detailed derivation process accordingly. We consider an $n \times n$-dimensional symmetric QUBO matrix $Q = \{q_{ij}\}_{n \times n}$ where $q_{ij} = q_{ji}, i = 1..n, j = 1..n$.

$$Q = \begin{bmatrix} q_{11} & q_{12} & \dots & q_{1n} \\ q_{21} & q_{22} & & q_{2n} \\ & \vdots & \ddots & \vdots \\ q_{n1} & q_{n2} & \cdots & q_{nn} \end{bmatrix} \tag{10}$$

The current QUBO solution $x = \{x_1, x_2, ..., x_n\}$ is an $n$-dimensional vector. $N = \{1, 2, ..., n\}$ indexes the elements of $x$ and both the rows and columns of $Q$. Thus, Equation (1) is expanded as follows.

$$f(x) = \sum_{i=1}^{n} q_{ii}x_i{}^2 + \sum_{i=1}^{n}\sum_{j=1}^{i-1} q_{ij}x_i x_j + \sum_{i=1}^{n}\sum_{j=i-1}^{n} q_{ij}x_i x_j \tag{11}$$

We examine the effect of flipping the $k$-th variable on the solution, yielding a new solution $x'$ in which $x'_i$ is defined by:

$$x'_i = \begin{cases} 1 - x_i, i = k \\ x_i, i \neq k \end{cases} \tag{12}$$

The $OFV$ for $x'$ is then determined.

$$f(x') = \sum_{i=1}^{n} q_{ii}x'_i{}^2 + \sum_{i=1}^{n}\sum_{j=1}^{i-1} q_{ij}x'_i x'_j + \sum_{i=1}^{n}\sum_{j=i-1}^{n} q_{ij}x'_i x'_j \tag{13}$$

Subsequently, the OFI of the variable $x_k$ is calculated as follows:

$$OFI(x_k) = f(x') - f(x) = (x'_k - x_k)$$

$$\left[ q_{kk}(x'_k + x_k) \quad + \sum_{i \in N, i \neq k} q_{ik}x_i + \sum_{j \in N, j \neq k} q_{kj}x_j \right] \tag{14}$$

Because of $x'_k + x_k = 1$ and $q_{ij} = q_{ji}$, the function simplifies to:

$$OFI(x_k) = (x'_k - x_k)\left[ q_{kk} + 2\sum_{i \in N, i \neq k} q_{ik}x_i \right] \tag{15}$$

In this case, we define the sum of the $k$-th row $Q_k^{sum}$ as follows:

$$Q_k^{sum} = \sum_{j \in N} q_{kj}x_j \tag{16}$$

All the $Q^{sum}$ across all variables can be calculated as follows:

$$Q^{sum} = Q \cdot x \tag{17}$$

Then, the flip of $OFI(x_k)$ can be divided into two cases: 0 to 1 and 1 to 0. In the first case where $x_k = 0$ and $x'_k = 1$, the difference $x'_k - x_k = 1$, which modifies Equation (15) to:

$$OFI(x_k) = q_{kk} + 2\sum_{j \in N, j \neq k} q_{kj}x_j \tag{18}$$

Since $x_k = 0$, we have $q_{kk}x_k = 0$, thereby expanding $OFI(x_k)$ as:

$$\begin{aligned} OFI(x_k) &= q_{kk} + 2\sum_{j \in N, j \neq k} q_{kj}x_j \\ &= q_{kk} + 2\sum_{j \in N, j \neq k} q_{kj}x_j + 2q_{kk}x_k \\ &= q_{kk} + 2Q_k^{sum} \end{aligned} \tag{19}$$

Conversely, when $x_k = 1$ and $x'_k = 0$, the difference $x'_k - x_k = -1$, adjusting Equation (15) to:

$$OFI(x_k) = -q_{kk} - 2\sum_{j \in N, j \neq k} q_{kj}x_j \tag{20}$$

Because of $q_{kk}x_k = q_{kk}$ when $x_k = 1$. The $OFI(x_k)$ can be expanded as follows.

$$\begin{aligned} OFI(x_k) &= -q_{kk} - 2\sum_{j \in N, j \neq k} q_{kj}x_j \\ &= q_{kk} - 2\sum_{j \in N} q_{kj}x_j \\ &= q_{kk} - 2Q_k^{sum} \end{aligned} \tag{21}$$

Hence, the calculation of $OFI(x_k)$ can be concluded as follow:

$$OFI(x_k) = \begin{cases} q_{kk} + 2Q_k^{sum}, & x_k = 0 \\ q_{kk} - 2Q_k^{sum}, & x_k = 1 \end{cases} \tag{22}$$

Finially, we define the diagonal elements of the matrix $Q$ as $Q^{diag} \in \mathbb{R}^{N \times 1}$ and propose the OFI batch calculation as follows:

$$OFI(x) = Q^{diag} + 2(1 - 2x)Q^{sum} \tag{23}$$

We evaluate the effectiveness of the proposed batch calculation of OFI using the following Greedy Flip method.

# C Batch Greedy Flip Algorithm

During the MDP of QUBO, we execute the flip with the highest positive OFI at each state in a greedy manner to derive the Greedy Flip algorithm. In Figure 12, we provide a comprehensive illustration of the Greedy Flip algorithm's execution within a specific QUBO instance comprising four variables. Notably, variable $x_1$ undergoes two repeated flips.

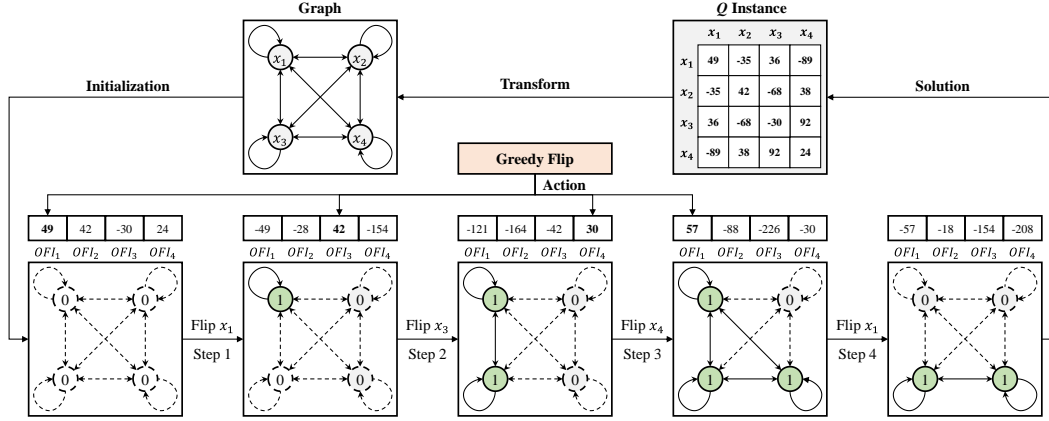

Figure 12: The Greedy Flip algorithm of QUBO.

Enhanced by the proposed OFI batch calculation method, we propose the Batch Greedy Flip (BGF) method for supporting the proposed GST. Besides, BGF is not only used as a competitor to validate the effectiveness of VCM but also as an enhancement component for VCM to integrate a powerful hybrid construction method VCM-BGF. The pseudo-code of BGF is presented in Algorithm 2. It is worth noting that the length of the solution for the batch calculation is $N + 1$, consisting of the problem solution with a set-aside point. In case the OFI of all variables in the current solution is non-positive OFI, the value of this set-aside point will be flipped.

---

**Algorithm 2** The Batch Greedy Flip (BGF) algorithm

---

**Input:** the batch size $B$, the matrix of the batch $Q = \{Q_t | t = 1, ..., B\}$, $Q_t \in \mathbb{R}^{(N+1) \times (N+1)}$, the initial solution of the batch $x = \{x_t | t = 1, ..., B\}$, $x_t \in \mathbb{R}^{(N+1) \times 1}$
**Output:** the improved solution $x^*$
Calculate $Q^{diag} \in \mathbb{R}^{B \times (N+1) \times 1}$
**while** True **do**
    $OFI_t \leftarrow Q_t^{diag} + 2(1 - 2x_t)Q_t^{sum}$
    **if** $\forall OFI_{ti} \leq 0, t = \{1, ..., B\}, i = \{1, ..., N\}$ **then**
      $break$
    **else**
      **for** $t = 1$ **to** $B$ **do**
        **if** $\forall OFI_{ti} \leq 0, i = \{1, ..., N\}$ **then**
          $k \leftarrow N + 1$
        **else**
          $k \leftarrow \arg\max_{k \in \{1, ..., N\}} OFI_{tk}$
        **end if**
        $x_{tk} = 1 - x_{tk}$
      **end for**
    **end if**
**end while**

---

## C.1    The Efficiency of Batch Calculation

To investigate the effectiveness of the proposed OFI batch calculation, we conduct tests on 1,000 generated G instances with 20, 50, 100, 200, 500, and 1,000 variables, using BGF with batch sizes of 1, 32, 64, 128, and 256. For comparative validation, we apply the Traditional Greedy Flip (TGF) method as the competitor, which calculates the OFI in a sequential and instance-by-instance way. We plot the running time curve in Figure 13 and detail the data in Table 2. The results reveal that the OFI batch calculation can significantly enhance the computing efficiency of the Greedy Flip method, which brings orders of magnitude efficiency improvement. With a batch size of 1, BGF-01 operates similarly to TGF, calculating instance by instance, where the difference between these two methods lies in the OFI calculation process of all variables in each decision step. Based on the batch calculation, BGF distinguishes itself by calculating the OFI for all variables simultaneously within a single step, as opposed to the sequential approach of TGF. This capability affords BGF a marked efficiency edge which grows with problem size, exemplified by its requirement of only 0.696% of the traditional calculation time for G1000 instances. The efficiency gains from OFI batch calculation are even more pronounced when dealing with batch instance computing. This improvement stems from the lower computational complexity afforded by the matrix computation process and the GPU acceleration. Consequently, the proposed OFI batch calculation method represents a potent computational tool for QUBO research, with promising applications beyond the scope of this work to potentially expedite a broad range of existing QUBO methods.

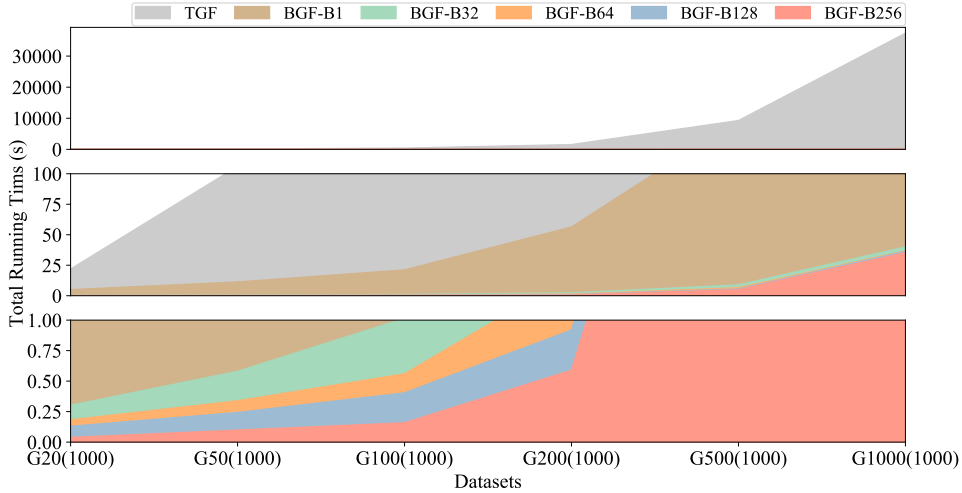

Figure 13: Total running time (s) of TGF and BGFs in different instances.

Table 2: Total running time of TGF and BGFs in different instances.

| INSTANCE | TGF | BGF-B1 | BGF-B32 | BGF-B64 | BGF-B128 | BGF-B256 |
|---|---|---|---|---|---|---|
| | TIME (S) | TIME (S) | TIME (S) | TIME (S) | TIME (S) | TIME (S) |
| G20(1000) | 21.92 | 5.01 | 0.30 | 0.18 | 0.13 | **0.04** |
| G50(1000) | 106.08 | 11.34 | 0.58 | 0.34 | 0.24 | **0.10** |
| G100(1000) | 428.49 | 21.20 | 1.02 | 0.56 | 0.40 | **0.16** |
| G200(1000) | 1549.55 | 56.41 | 2.38 | 1.37 | 0.92 | **0.59** |
| G500(1000) | 9283.51 | 144.51 | 8.94 | 6.62 | 5.66 | **5.10** |
| G1000(1000) | 37365.85 | 260.12 | 40.04 | 36.87 | 36.42 | **35.32** |

## C.2    BGF with Different Initial Solution

To investigate the effect of the initial solution on the BGF, we apply various initial solutions with different percentages of 1, including 0, 25%, 50%, 75%, and 100%. We set the test batch size to 256 and take the generated instances for testing, including G20(1000), G50(1000), G100(1000),

G200(1000), G500(1000), and G1000(1000). Except for 0 and 100%, all BGFs are run ten times on average. The results of average $OFV$ and total running time are shown in Table 3, where we can find that the BGF with 100% of 1 outperforms all other test sizes in terms of both quality and speed. Therefore, we take the BGF-100% as the default BGF.

Table 3: Average $OFV$ and total running time of BGF with different initial solutions.

| INSTANCE | BGF-0 | | BGF-25% | | BGF-50% | | BGF-75% | | BGF-100% | |
|---|---|---|---|---|---|---|---|---|---|---|
| | $OFV$ | TIME (S) | $OFV$ | TIME (S) | $OFV$ | TIME (S) | $OFV$ | TIME (S) | $OFV$ | TIME (S) |
| G20(1000) | 2838 | 0.09 | 2841 | 0.06 | 2859 | 0.05 | 2888 | 0.05 | **2916** | **0.04** |
| G50(1000) | 11659 | 0.18 | 11687 | 0.13 | 11736 | 0.12 | 11800 | 0.11 | **11901** | **0.10** |
| G100(1000) | 33381 | 0.34 | 33460 | 0.27 | 33522 | 0.26 | 33679 | 0.22 | **33892** | **0.16** |
| G200(1000) | 94907 | 1.21 | 95036 | 1.04 | 95251 | 0.91 | 95556 | 0.76 | **96022** | **0.59** |
| G500(1000) | 379096 | 11.75 | 379624 | 9.74 | 380133 | 8.16 | 380816 | 6.65 | **382412** | **5.10** |
| G1000(1000) | 1075398 | 88.50 | 1076710 | 71.81 | 1077693 | 59.09 | 1079293 | 48.13 | **1082420** | **35.32** |

## D   Wilcoxon Signed-rank Tests

We perform Wilcoxon signed-rank tests on the gap results obtained from the trained VCMs in comparison to other algorithms across 31 instances from the B and P sets. The results are summarized in Table 4, which clearly show significant differences between the VCM results and those of other algorithms, with the highest P-value being 3.09E-03 between VCM100 and BGF.

Table 4: The P-value results of Wilcoxon signed-rank test between VCMs and other algorithms.

| ALGORITHM | VCM10 | VCM20 | VCM50 | VCM100 |
|---|---|---|---|---|
| DIAG | 1.17E-06 | 1.30E-06 | 4.53E-06 | 4.97E-06 |
| SR | 1.77E-05 | 1.43E-06 | 5.46E-06 | 2.56E-06 |
| BGF | 1.17E-06 | 1.94E-05 | 2.12E-04 | 3.09E-03 |
| DRLH-B | 1.17E-06 | 1.17E-06 | 1.17E-06 | 1.17E-06 |
| S2V-DQN-B | 3.79E-06 | 3.79E-06 | 4.23E-06 | 3.79E-06 |
| ECO-DQN-B | 3.79E-06 | 3.79E-06 | 3.79E-06 | 3.79E-06 |
| VCM10-BGF | 1.17E-06 | 1.17E-06 | 1.30E-06 | 2.56E-06 |
| VCM20-BGF | 2.56E-06 | 1.17E-06 | 2.56E-06 | 1.43E-06 |
| VCM50-BGF | 1.17E-06 | 1.17E-06 | 1.17E-06 | 1.17E-06 |
| VCM100-BGF | 1.74E-06 | 1.17E-06 | 1.74E-06 | 1.17E-06 |
| PI-GNN(2-LAYER) | 1.17E-06 | 1.17E-06 | 1.17E-06 | 1.17E-06 |
| PI-GNN(3-LAYER) | 1.17E-06 | 1.17E-06 | 1.17E-06 | 1.17E-06 |
| PI-GNN(5-LAYER) | 1.17E-06 | 1.17E-06 | 1.17E-06 | 1.17E-06 |
| GUROBI-1S | 1.17E-06 | 1.17E-06 | 1.17E-06 | 1.17E-06 |
| GUROBI-1H | 1.17E-06 | 1.17E-06 | 1.17E-06 | 1.17E-06 |
| VCM-BGF-HB | 2.52E-05 | 1.30E-06 | 1.30E-06 | 1.17E-06 |

## E   Ablation Study

To validate the rationally designed network architecture of our VCM, we test the training performance of VCM by selectively ablating specific modules. We employ the VCM-() with the ablation module to distinguish different VCMs. Specifically, we select the VCM-(ReLU), VCM-(tanh1), VCM-(tanh2), VCM-(M1) and VCM-(M2) for validation. The ablation module can be quickly queried in Figure 9. The corresponding impacts of these ablations are captured in Figure 14, illustrating that any deficient module degrades VCM performance, notably ReLU and tanh2 modules. ReLU is pivotal for extracting value features, while tanh2 normalizes value features within a range of (-1, 1), essential for the unification of DVN extraction.

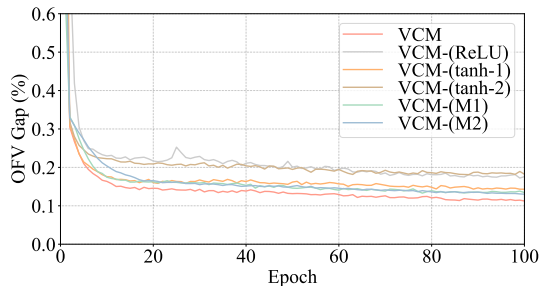

Figure 14: The VCM training process of ablation study at instance size 50.

# F Parameter Study

## F.1 The Scaling Parameter $\alpha$

We train the VCM with $\alpha$ of 2, 4 (default), 6, and None (without scaling function) at instance size 50. The training process is shown in Figure 15, where we observe that the $\alpha$ can improve the performance of the VCM. Notably, the default VCM ($\alpha = 4$) achieves most of the training lead. We believe the scaling function of data compression makes tanh-1 more sensitive to input processing in the early stages. According to Equation (9), the optimal $\alpha = 4$ can compress the first input value (sum of each row) of DVN to an average of about 2, regardless of instance size and dataset distribution. It should be noted that when the input value exceeds 2, there is a negligible difference in the output by tanh. Therefore, using tanh-1 in the first depth of the default VCM can better compress and distinguish features, thereby accelerating model learning.

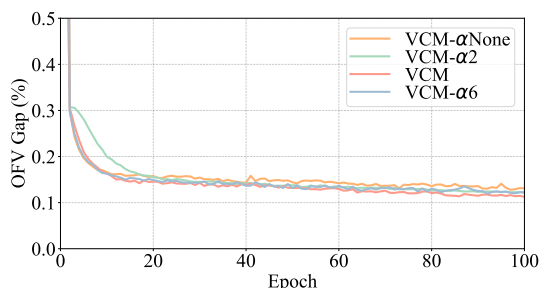

Figure 15: Training process of VCM with different $\alpha$ at instance size 50.

## F.2 The DVN Depth $d$

A certain depth of DVN extraction is mandatory for ensuring interaction and coherence between the row and column features. To assess the impact of varying depths $d$ on the performance of VCM, we set the training $d$ of VCM50 to 10, 20, 30, 40 (default), 50, and 100 within an instance size of 50 for validation. As depicted in Figure 16 in Appendix E, the training trajectories of these six VCMs demonstrate that increasing depth (within our test range) enhances feature extraction precision, yielding a consistent improvement in VCM performance. Notably, the performance gain is pronounced as the depth extends from 10 to 30, but it plateaus thereafter. This trend suggests that a certain threshold of DVN depth is required to accomplish thorough value feature extraction from $Q$. While deeper $d$ can enhance performance, the training costs also incur greater. Therefore, we elected to set $d$ at the most cost-efficient value of 40 by default.

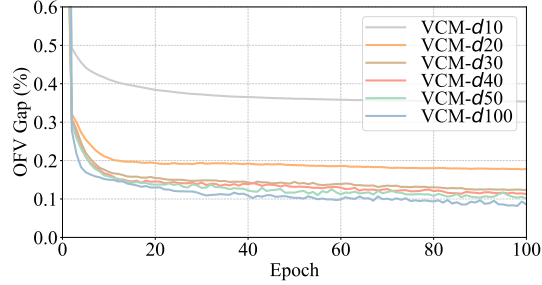

Figure 16: Training process of VCM with DVN depth $d$ at instance size 50.

### F.3 The Value Feature Size $h$

To evaluate the impact of $h$ on VCM performance, we train the model with various values of $h$ =32, 64, 128 (default), and 256 for an instance size of 50. As shown in Figure 17, the results indicate a significant improvement in VCM performance with the increase of $h$. Notably, the default setting of $h = 128$ achieves the highest training advancement, indicating that increasing the value feature size within a reasonable range enhances model performance effectively.

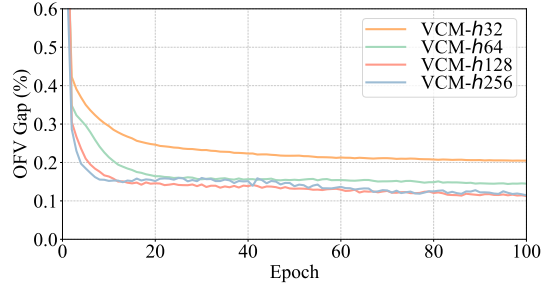

Figure 17: Training process of VCM with value feature size $h$ at instance size 50.

## G   Results of Size Generalization

This section details the results of the size generation experiment. We conduct tests on an additional 1,000 generated G instances with 20, 50, 100, 200, 500, and 1,000 variables. The VCM-BGF-HB is applied as the current optimal baseline for gap calculation. The detailed results are depicted in Table 5. Notably, Gurobi incurs a significant time cost, and the results for G set with over 100 variables are omitted as each instance takes more than 1 hour.

## H   Results of Distribution Generalization

This section details the results of the distribution generation experiment in the Model Study. We conduct tests with five different distributions on these G instances with 20, 50, and 100 variables using our trained corresponding-sized VCMs, including the standard normal random (RN-1) and the uniform random with probabilities of 0 for 10% (R-0.1), 30% (R-0.3), 60% (R-0.6), 90% (R-0.9). For each test dataset, we generate 1,000 G instances and also apply VCM-BGF-HB as the current optimal baseline to measure the gap. The results are detailed in Table 6, with heatmaps provided in Figure 18.

Table 5: Results on generated datasets.

| METHOD | G20(1000) | | G50(1000) | | G100(1000) | | G200(1000) | | G500(1000) | | G1000(1000) | |
|---|---|---|---|---|---|---|---|---|---|---|---|---|
| | OBJECTIVE FUNCTION VALUE BASELINE | | | | | | | | | | | |
| | 2965.095 | | 12070.455 | | 34350.38 | | 97260.38 | | 386638.40 | | 1093972.99 | |
| | GAP (%) | ART (MS) | GAP (%) | ART (MS) | GAP (%) | ART (MS) | GAP (%) | ART (MS) | GAP (%) | ART (MS) | GAP (%) | ART (MS) |
| DIAG | 84.098 | 0.1 | 90.229 | 0.1 | 92.569 | 0.1 | 94.882 | 0.2 | 96.561 | 0.4 | 97.632 | 0.9 |
| SR | 31.637 | 3.5E-02 | 32.594 | 3.9E-02 | 32.628 | 3.5E-02 | 33.189 | 4.5E-02 | 32.915 | 4.6E-02 | 33.054 | 4.6E-02 |
| VCM10 | **0.060** | 4.4 | **0.108** | 4.4 | **0.151** | 4.8 | 0.248 | 7.4 | **0.338** | 7.4 | **0.414** | 7.5 |
| VCM20 | 0.121 | 4.4 | 0.112 | 4.7 | 0.196 | 4.7 | 0.257 | 7.3 | 0.409 | 7.4 | 0.507 | 7.5 |
| VCM50 | 0.200 | 4.4 | 0.129 | 4.4 | 0.167 | 4.8 | 0.230 | 7.3 | 0.382 | 7.4 | 0.523 | 7.5 |
| VCM100 | 0.244 | 4.4 | 0.117 | 4.4 | 0.207 | 4.8 | **0.207** | 7.3 | 0.357 | 7.4 | 0.508 | 7.5 |
| BGF | 1.642 | 5.2 | 1.401 | 11.7 | 1.335 | 23.1 | 1.273 | 60.8 | 1.093 | 158.1 | 1.056 | 294.2 |
| DRLH-B | 2.034 | 8.4 | 1.871 | 17.8 | 1.848 | 34.3 | 1.926 | 91.5 | 1.930 | 220.4 | 1.914 | 397.3 |
| ECO-DQN-B | 0.321 | 43.2 | 0.362 | 109.5 | 0.471 | 216.6 | 0.731 | 636.2 | 1.004 | 2.3E+03 | 1.160 | 1.3E+04 |
| S2V-DQN-B | 4.007 | 22.9 | 4.966 | 57.1 | 5.758 | 115.0 | 6.414 | 338.5 | 7.173 | 1.2E+03 | 7.567 | 6.4E+03 |
| VCM10-BGF | 0.025 | 5.1 | 0.058 | 5.2 | 0.089 | 5.7 | 0.148 | 9.3 | 0.209 | 11.6 | 0.258 | 16.5 |
| VCM20-BGF | **0.010** | 5.1 | 0.037 | 5.5 | 0.066 | 5.7 | 0.107 | 9.3 | 0.173 | 12.6 | 0.223 | 18.8 |
| VCM50-BGF | 0.036 | 5.1 | **0.019** | 5.2 | 0.044 | 5.7 | 0.064 | 9.3 | 0.113 | 12.4 | 0.163 | 19.6 |
| VCM100-BGF | 0.052 | 5.1 | 0.027 | 5.1 | **0.042** | 5.7 | **0.050** | 9.2 | **0.094** | 12.3 | **0.138** | 19.6 |
| GUROBI | 0* | 54.3 | 0* | 3.7E+03 | - | >1H | - | - | - | - | - | - |
| VCM-BGF-HB | 0.0003 | 509.1 | 0.0005 | 526.0 | 0* | 572.3 | 0* | 926.6 | 0* | 1.2E+03 | 0* | 1.9E+03 |

[1] The **Bold** indicates the best average result in different classes of methods.
[2] The * indicates the provider of the applied $OFV$ baseline.

Table 6: Distribution generalization results on generated datasets.

| SIZE | INSTANCE METHOD | G-RN-1 $OFV$ | GAP (%) | ART (MS) | G-R-0.1 $OFV$ | GAP (%) | ART (MS) | G-R-0.3 $OFV$ | GAP (%) | ART (MS) | G-R-0.6 $OFV$ | GAP (%) | ART (MS) | G-R-0.9 $OFV$ | GAP (%) | ART (MS) |
|---|---|---|---|---|---|---|---|---|---|---|---|---|---|---|---|---|
| 20 | BGF | 5089.9 | 1.34 | 5.9 | 793.9 | 1.32 | 3.7 | 1560.5 | 1.74 | 4.7 | 2234.4 | 1.55 | 4.7 | 2770.7 | 1.49 | 4.8 |
| | VCM | 5154.2 | 0.09 | 4.7 | 803.4 | 0.14 | 4.4 | 1586.8 | 0.08 | 4.4 | 2266.6 | 0.13 | 4.4 | 2809.2 | 0.12 | 4.3 |
| | VCM-BGF | **5157.8** | **0.02** | 5.4 | **804.4** | **0.02** | 5.1 | **1587.9** | **0.01** | 5.1 | **2269.0** | **0.02** | 5.1 | **2811.9** | **0.02** | 5.1 |
| | VCM-BGF-HB | 5158.9* | - | 2.2E+03 | 804.5* | - | 2.0E+03 | 1588.1* | - | 2.0E+03 | 2269.5* | - | 2.0E+03 | 2812.5* | - | 2.0E+03 |
| 50 | BGF | 20457.5 | 1.40 | 12.3 | 3557.7 | 1.64 | 10.1 | 6392.9 | 1.64 | 10.6 | 9121.7 | 1.51 | 10.8 | 11280.1 | 1.36 | 11.1 |
| | VCM | 20724.8 | 0.11 | 4.6 | 3611.1 | 0.17 | 4.4 | 6495.0 | 0.07 | 4.4 | 9247.4 | 0.15 | 4.4 | 11421.5 | 0.12 | 4.4 |
| | VCM-BGF | **20743.0** | **0.02** | 5.5 | **3616.2** | **0.02** | 5.2 | **6498.4** | **0.02** | 5.1 | **9259.3** | **0.03** | 5.1 | **11433.4** | **0.02** | 5.1 |
| | VCM-BGF-HB | 20747.9* | - | 2.2E+03 | 3617.1* | - | 2.1E+03 | 6499.8* | - | 2.0E+03 | 9261.6* | - | 2.1E+03 | 11435.5* | - | 2.1E+03 |
| 100 | BGF | 58031.7 | 1.38 | 23.3 | 10505.5 | 1.64 | 20.7 | 18462.2 | 1.48 | 24.4 | 26199.1 | 1.47 | 20.9 | 32279.1 | 1.27 | 22.4 |
| | VCM | 58756.3 | 0.15 | 5.0 | 10664.7 | 0.15 | 4.8 | 18713.0 | 0.14 | 4.8 | 26556.2 | 0.13 | 5.2 | 32641.2 | 0.16 | 4.7 |
| | VCM-BGF | **58824.2** | **0.04** | 6.0 | **10676.8** | **0.04** | 5.6 | **18732.2** | **0.04** | 5.7 | **26584.1** | **0.03** | 6.2 | **32683.1** | **0.03** | 5.7 |
| | VCM-BGF-HB | 58844.9* | - | 2.4E+03 | 10680.6* | - | 2.2E+03 | 18739.1* | - | 2.3E+03 | 26591.2* | - | 2.5E+03 | 32693.5* | - | 2.3E+03 |

[1] The **Bold** indicates the best average result in different datasets.
[2] The * indicates the provider of the applied $OFV$ baseline.

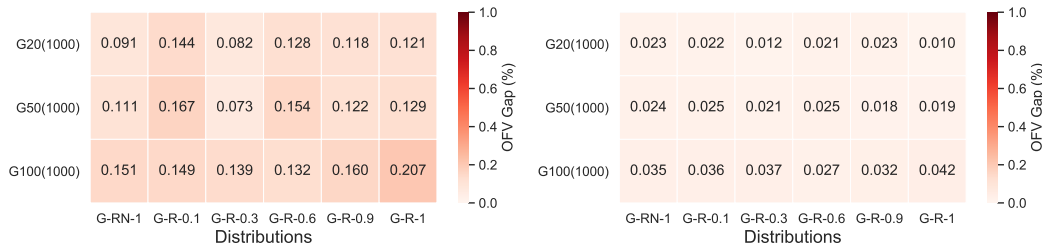

(a) The average gap in % of VCMs.        (b) The average gap in % of VCM-BGFs.

Figure 18: Heatmaps of distribution generalization obtained by VCMs under G-R-1 with corresponding instance size.

# I  Results of MaxCut Benchmarks

We extend our evaluation to MaxCut benchmarks, with the results presented below. Utilizing data from [15], we compile the results shown in Table 7, using the average objective function value as

the key indicator. Additionally, based on data from [51], we present the results in Table 8, where the average approximation ratios serve as the performance indicator.

The results confirm that VCM outperforms other methods, consistent with the advantages demonstrated in our test instances. This validates the applicability and robustness of our VCM in problems that can be recast as QUBO across different scales and distributions.

Table 7: Results on MaxCut benchmarks.

| INSTANCE | OPT | S2V-DQN | VCM50-$d$100 |
|---|---|---|---|
| G54100-G5410000 (10 INSTANCES) | 110.6 | 108.2 | **109.6(5MS)** |

Table 8: Results on large MaxCut benchmarks.

| INSTANCE | TABU | SOFTTABU | S2V-DQN | ECO-DQN | VCM-$d$100 | VCM-BGF-$d$100 |
|---|---|---|---|---|---|---|
| G32-G34 (2000 NODES) | 0.915 | 0.983 | 0.923 | 0.969 | 0.990(16MS) | **0.991(23MS)** |

## J    Results of DVN depths

This section details the results of the DVN depth experiment in the Model Study. We train VCM50s across various depths $d$ including 10, 20, 30, 40, 50, and 100. Each trained specific-size VCM is tested across an extended range of depths, including 10, 20, 30, 40, 50, 100, 200, and 300. The average gap (GAP) and average running time (ART) on benchmarks B2500(10) are listed in Table 9. The corresponding heatmap is illustrated in Figure 19.

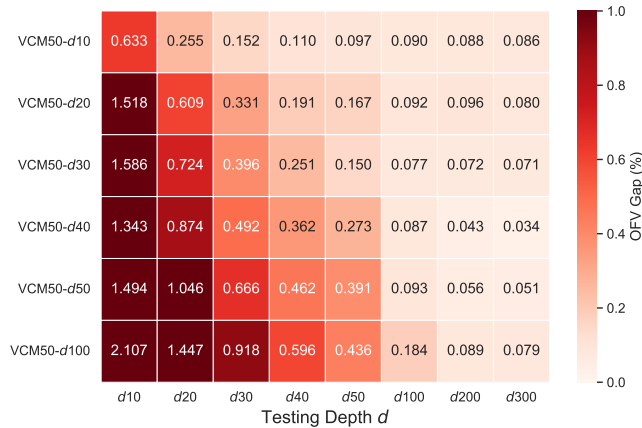

Figure 19: Heatmaps of test depths under VCM50 models with different training depths $d$.

Table 9: Results of DVN depths on benchmarks B2500(10).

| METHOD | 10 | | 20 | | 30 | | 40 | | 50 | | 100 | | 200 | | 300 | |
|---|---|---|---|---|---|---|---|---|---|---|---|---|---|---|---|---|
| | GAP (%) | ART (MS) | GAP (%) | ART (MS) | GAP (%) | ART (MS) | GAP (%) | ART (MS) | GAP (%) | ART (MS) | GAP (%) | ART (MS) | GAP (%) | ART (MS) | GAP (%) | ART (MS) |
| VCM50-$d$10 | 0.633 | 2.6 | 0.255 | 4.4 | 0.152 | 6.1 | 0.110 | 7.8 | 0.097 | 9.6 | 0.090 | 18.2 | 0.088 | 35.5 | 0.086 | 53.1 |
| VCM50-$d$20 | 1.518 | 2.6 | 0.609 | 4.4 | 0.331 | 6.0 | 0.191 | 7.8 | 0.167 | 9.6 | 0.092 | 18.3 | 0.096 | 35.5 | 0.080 | 53.1 |
| VCM50-$d$30 | 1.586 | 2.6 | 0.724 | 4.3 | 0.396 | 6.1 | 0.251 | 7.8 | 0.150 | 9.6 | 0.077 | 18.3 | 0.072 | 35.5 | 0.071 | 53.2 |
| VCM50-$d$40 | 1.343 | 2.6 | 0.874 | 4.4 | 0.492 | 6.1 | 0.362 | 7.8 | 0.273 | 9.6 | 0.087 | 18.3 | 0.043 | 35.5 | **0.034** | 52.8 |
| VCM50-$d$50 | 1.494 | 2.6 | 1.046 | 4.3 | 0.666 | 6.1 | 0.462 | 7.8 | 0.391 | 9.5 | 0.093 | 18.4 | 0.056 | 35.5 | 0.051 | 52.8 |
| VCM50-$d$100 | 2.107 | 2.6 | 1.447 | 4.3 | 0.918 | 6.1 | 0.596 | 7.8 | 0.436 | 9.6 | 0.184 | 18.6 | 0.089 | 35.5 | 0.079 | 52.8 |

In addition, we evaluate the VCM50 and VCM50-BGF at the default testing depth of $d = 40$ and a deeper depth of $d = 300$. The results on benchmarks and large well-known instances are presented in Table 10. We use Gurobi-1s as the baseline for comparison.

Table 10: Results on benchmarks and large well-known instances.

| ALGORITHM | B2500(10) | | P3000(5) | | P4000(5) | | P5000(5) | | P6000(3) | | P7000(3) | |
|---|---|---|---|---|---|---|---|---|---|---|---|---|
| | OBJECTIVE FUNCTION VALUE BASELINE (OPTIMAL) | | | | | | | | | | | |
| | 1479921.4 | | 5134727.2 | | 7869134.2 | | 10973791.4 | | 13950582.0 | | 17725010.33 | |
| | GAP (%) | ART (MS) | GAP (%) | ART (MS) | GAP (%) | ART (MS) | GAP (%) | ART (MS) | GAP (%) | ART (MS) | GAP (%) | ART (MS) |
| GUROBI-1S | 0.034 | 1.0E+03 | 0.070 | 1.0E+03 | 0.108 | 1.0E+03 | 0.091 | 1.0E+03 | 0.122 | 1.0E+03 | 0.145 | 1.0E+03 |
| VCM50 | 0.362 | 8 | 0.861 | 8 | 0.669 | 8 | 0.783 | 8 | 0.806 | 9 | 0.702 | 9 |
| VCM50-$d$300 | 0.034 | 52 | 0.066 | 53 | 0.115 | 52 | 0.099 | 53 | 0.144 | 76 | 0.144 | 116 |
| VCM50-BGF | 0.136 | 44 | 0.277 | 100 | 0.214 | 170 | 0.262 | 326 | 0.267 | 523 | 0.224 | 770 |
| VCM50-$d$300-BGF | **0.027** | 64 | **0.04** | 79 | **0.088** | 108 | **0.078** | 145 | **0.109** | 249 | **0.108** | 331 |

## K    Results of Degradation in GCN

We apply GCN [42] to replace the DVN in VCM and confirm performance degradation with increased layers, consistent with [42]. We train GCN and VCM by GST on size 50 under the same settings. The training details are shown in Figure 20 and the optimal training gaps are shown in Figure 21.

Obviously, the GCN suffers from performance degradation, which is consistent with the conclusion in [42]. However, the performance of VCM steadily improves with increasing depth. Besides, the neural parameters in GCN layers are independent, whereas neural units in VCM depth are consistent, resulting in lower training costs under the same neural settings.

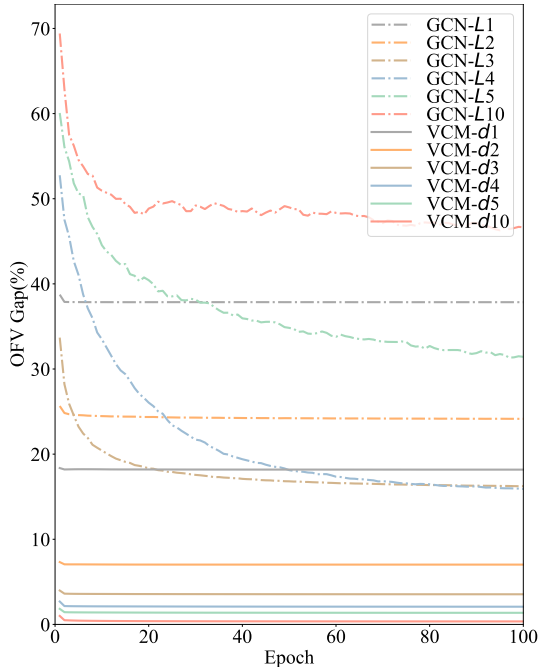

Figure 20: The training processes of GCN and VCM under different training layer $L$ or depth $d$ at instance size 50.

## L    Results of Very Large Instances

This section presents the results obtained from testing very large instances. We conduct experiments on a single generated G instance with sizes of 10,000 and 20,000, using distributions R-0.1 and R-0.3.

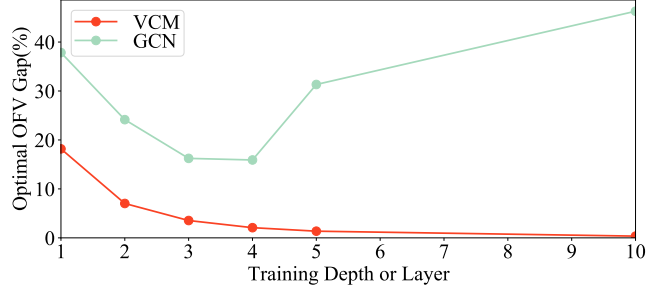

Figure 21: The optimal training OFV gap of GCN and VCM under different training layer $L$ or depth $d$ at instance size 50.

The model VCM50-$d$300 is evaluated against Gurobi, which served as the baseline with a maximum allowable runtime of 1 second, referred to as Gurobi-1s.

To assess performance, we utilize the OFV gap between Gurobi-1s and VCM50-$d$300 as the metric. The results are summarized in Table 11. Notably, VCM50-$d$300 consistently outperforms Gurobi-1s in terms of both solution quality and execution speed, demonstrating its effectiveness on very large instances across various distributions.

Table 11: Results on generated large size instances.

| METHOD | G10000-R0.1(1) | | G10000-R0.3(1) | | G20000-R0.1(1) | | G20000-R0.3(1) | |
|---|---|---|---|---|---|---|---|---|
| | GAP (%) | T (MS) | GAP (%) | T (MS) | GAP (%) | T (MS) | GAP (%) | T (MS) |
| GUROBI-1S | 0.183 | 1.0E+03 | 0.045 | 1.0E+03 | 0.091 | 1.0E+03 | 0.108 | 1.0E+03 |
| VCM50-$d$300 | **0** | 169 | **0** | 172 | **0** | 637 | **0** | 676 |

